# Learning Image Descriptors with the Boosting-Trick

**Tomasz Trzcinski, Mario Christoudias, Vincent Lepetit and Pascal Fua**
CVLab, EPFL, Lausanne, Switzerland
`firstname.lastname@epfl.ch`

## Abstract

In this paper we apply boosting to learn complex non-linear local visual feature representations, drawing inspiration from its successful application to visual object detection. The main goal of local feature descriptors is to distinctively represent a salient image region while remaining invariant to viewpoint and illumination changes. This representation can be improved using machine learning, however, past approaches have been mostly limited to learning linear feature mappings in either the original input or a kernelized input feature space. While kernelized methods have proven somewhat effective for learning non-linear local feature descriptors, they rely heavily on the choice of an appropriate kernel function whose selection is often difficult and non-intuitive. We propose to use the *boosting-trick* to obtain a non-linear mapping of the input to a high-dimensional feature space. The non-linear feature mapping obtained with the boosting-trick is highly intuitive. We employ gradient-based weak learners resulting in a learned descriptor that closely resembles the well-known SIFT. As demonstrated in our experiments, the resulting descriptor can be learned directly from intensity patches achieving state-of-the-art performance.

## 1 Introduction

Representing salient image regions in a way that is invariant to unwanted image transformations is a crucial Computer Vision task. Well-known local feature descriptors, such as the Scale Invariant Feature Transform (SIFT) [1] or Speeded Up Robust Features (SURF) [2], address this problem by using a set of hand-crafted filters and non-linear operations. These descriptors have become prevalent, even though they are not truly invariant with respect to various viewpoint and illumination changes which limits their applicability.

In an effort to address these limitations, a fair amount of work has focused on learning local feature descriptors [3, 4, 5] that leverage labeled training image patches to learn invariant feature representations based on local image statistics. Although significant progress has been made, these approaches are either built on top of hand-crafted representations [5] or still require significant parameter tuning as in [4] which relies on a non-analytical objective that is difficult to optimize.

Learning an invariant feature representation is strongly related to learning an appropriate similarity measure or metric over intensity patches that is invariant to unwanted image transformations, and work on descriptor learning has been predominantly focused in this area [3, 6, 5]. Methods for metric learning that have been applied to image data have largely focused on learning a linear feature mapping in either the original input or a kernelized input feature space [7, 8]. This includes previous boosting-based metric learning methods that thus far have been limited to learning linear feature transformations [3, 7, 9]. In this way, non-linearities are modeled using a predefined similarity or kernel function that implicitly maps the input features to a high-dimensional feature space where the transformation is assumed to be linear. While these methods have proven somewhat effective for learning non-linear local feature mappings, choosing an appropriate kernel function is often non-intuitive and remains a challenging and largely open problem. Additionally, kernel methods involve

an optimization whose problem complexity grows quadratically with the number of training examples making them difficult to apply to large problems that are typical to local descriptor learning.

In this paper, we apply boosting to learn complex non-linear local visual feature representations drawing inspiration from its successful application to visual object detection [10]. Image patch appearance is modeled using local non-linear filters evaluated within the image patch that are effectively selected with boosting. Analogous to the kernel-trick, our approach can be seen as applying a *boosting-trick* [11] to obtain a non-linear mapping of the input to a high-dimensional feature space. Unlike kernel methods, the boosting-trick allows for the definition of intuitive non-linear feature mappings. Also, our learning approach scales linearly with the number of training examples making it more easily amenable to large scale problems and results in highly accurate descriptor matching.

We build upon [3] that also relies on boosting to compute a descriptor, and show how we can use it as a way to efficiently select features, from which we compute a compact representation. We also replace the simple weak learners of [3] by non-linear filters more adapted to the problem. In particular, we employ image gradient-based weak learners similar to [12] that share a close connection with the non-linear filters used in proven image descriptors such as SIFT and Histogram-of-Oriented Gradients (HOG) [13]. Our approach can be seen as a generalization of these methods cast within a principled learning framework. As seen in our experiments, our descriptor can be learned directly from intensity patches and results in state-of-the-art performance rivaling its hand-designed equivalents.

To evaluate our approach we consider the image patch dataset of [4] containing several hundreds of thousands of image patches under varying viewpoint and illumination conditions. As baselines we compare against leading contemporary hand-designed and learned local feature descriptors [1, 2, 3, 5]. We demonstrate the effectiveness of our approach on this challenging dataset, significantly outperforming the baseline methods.

## 2 Related work

Machine learning has been applied to improve both matching efficiency and accuracy of image descriptors [3, 4, 5, 8, 14, 15]. Feature hashing methods improve the storage and computational requirements of image-based features [16, 14, 15]. Salakhutdinov and Hinton [16, 17] develop a semantic hashing approach based on Restricted Boltzman Machines (RBMs) applied to binary images of digits. Similarly, Weiss *et al.* [14] present a spectral hashing approach that learns compact binary codes for efficient image indexing and matching. Kulis and Darrell [15] extend this idea to explicitly minimize the error between the original Euclidean and computed Hamming distances. Many of these approaches presume a given distance or similarity measure over a pre-defined input feature space. Although they result in efficient description and indexing in many cases they are limited to the matching accuracy of the original input space. In contrast, our approach learns a non-linear feature mapping that is specifically optimized to result in highly accurate descriptor matching.

Methods to metric learning learn feature spaces tailored to a particular matching task [5, 8]. These methods assume the presence of annotated label pairs or triplets that encode the desired proximity relationships of the learned feature embedding. Jain *et al.* [8] learn a Mahalanobis distance metric defined using either the original input or a kernelized input feature space applied to image classification and matching. Alternatively, Strecha *et al.* [5] employ Linear Discriminant Analysis to learn a linear feature mapping from binary-labeled example pairs. Both of these methods are closely related, offering different optimization strategies for learning a Mahalanobis-based distance metric. While these methods improve matching accuracy through a learned feature space, they require the presence of a pre-selected kernel function to encode non-linearities. Such approaches are well suited for certain image indexing and classification tasks where task-specific kernel functions have been proposed (e.g., [18]). However, they are less applicable to local image feature matching, for which the appropriate choice of kernel function is less understood.

Boosting has also been applied for learning Mahalanobis-based distance metrics involving high-dimensional input spaces overcoming the large computational complexity of conventional positive semi-definite (PSD) solvers based on the interior point method [7, 9]. Shen *et al.* [19] proposed a PSD solver using column generation techniques based on AdaBoost, that was later extended to involve closed-form iterative updates [7]. More recently, Bi *et al.* [9] devised a similar method exhibiting even further improvements in computational complexity with application to bio-medical imagery. While these methods also use boosting to learn a feature mapping, they have emphasized

computational efficiency only considering linear feature embeddings. Our approach exhibits similar computational advantages, however, has the ability to learn non-linear feature mappings beyond what these methods have proposed.

Similar to our work, Brown *et al.* [4] also consider different feature pooling and selection strategies of gradient-based features resulting in a descriptor which is both short and discriminant. In [4], however, they optimize on the combination of handcrafted blocks, and their parameters. The criterion they consider—the area below the ROC curve—is not analytical and thus difficult to optimize, and does not generalize well. In contrast, we provide a generic learning framework for finding such representations. Moreover, the form of our descriptor is much simpler. Simultaneous to this work, similar ideas were explored in [20, 21]. While these approaches assume a sub-sampled or course set of pooling regions to mitigate tractability, we allow for the discovery of more generic pooling configurations with boosting.

Our work on boosted feature learning can be traced back to the work of Dollár *et al.* [22] where they apply boosting across a range of different features for pedestrian detection. Our approach is probably most similar to the boosted Similarity Sensitive Coding (SSC) method of Shakhnarovich [3] that learns a boosted similarity function from a family of weak learners, a method that was later extended in [23] to be used with a Hamming distance. In [3], only linear projection based weak-learners were considered. Also, Boosted SSC can often yield fairly high-dimensional embeddings. Our approach can be seen as an extension of Boosted SSC to form low-dimensional feature mappings. We also show that the image gradient-based weak learners of [24] are well adapted to the problem. As seen in our experiments, our approach significantly outperforms Boosted SSC when applied to image intensity patches.

## 3    Method

Given an image intensity patch $\mathbf{x} \in \mathcal{R}^D$ we look for a descriptor of $\mathbf{x}$ as a non-linear mapping $H(\mathbf{x})$ into the space spanned by $\{h_i\}_{i=1}^M$, a collection of thresholded non-linear response functions $h_i(\mathbf{x}) : \mathcal{R}^D \to \{-1, 1\}$. The number of response functions $M$ is generally large and possibly infinite.

This mapping can be learned by minimizing the exponential loss with respect to a desired similarity function $f(\mathbf{x}, \mathbf{y})$ defined over image patch pairs

$$\mathcal{L} = \sum_{i=1}^{N} \exp(-l_i f(\mathbf{x}_i, \mathbf{y}_i)) \tag{1}$$

where $\mathbf{x}_i, \mathbf{y}_i \in \mathcal{R}^D$ are training intensity patches and $l_i \in \{-1, 1\}$ is a label indicating whether it is a similar $(+1)$ or dissimilar $(-1)$ pair.

The Boosted SSC method proposed in [3] considers a similarity function defined by a simply weighted sum of thresholded response functions

$$f(\mathbf{x}, \mathbf{y}) = \sum_{i=1}^{M} \alpha_i h_i(\mathbf{x}) h_i(\mathbf{y}) \,. \tag{2}$$

This defines a weighted hash function with the importance of each dimension $i$ given by $\alpha_i$.

Substituting this expression into Equation (1) gives

$$\mathcal{L}_{SSC} = \sum_{i=1}^{N} \exp\left(-l_i \sum_{j=1}^{M} \alpha_j h_j(\mathbf{x}_i) h_j(\mathbf{y}_i)\right) \,. \tag{3}$$

In practice $M$ is large and in general the number of possible $h_i$'s can be infinite making the explicit optimization of $\mathcal{L}_{SSC}$ difficult, which constitutes a problem for which boosting is particularly well suited [25]. Although boosting is a greedy optimization scheme, it is a provably effective method for constructing a highly accurate predictor from a collection of weak predictors $h_i$.

Similar to the kernel trick, the resulting *boosting-trick* also maps each observation to a high-dimensional feature space, however, it computes an explicit mapping for which the $\alpha_i$'s that define $f(\mathbf{x}, \mathbf{y})$ are assumed to be sparse [11]. In fact, Rosset *et al.* [26] have shown that under certain

settings boosting can be interpreted as imposing an $L_1$ sparsity constraint over the response function weights $\alpha_i$. As will be seen below, unlike the kernel trick, this allows for the definition of high-dimensional embeddings well suited to the descriptor matching task whose features have an intuitive explanation.

Boosted SSC employs linear response weak predictors based on a linear projection of the input. In contrast, we consider non-linear response functions more suitable for the descriptor matching task as discussed in Section 3.3. In addition, the greedy optimization can often yield embeddings that although accurate are fairly redundant and inefficient.

In what follows, we will present our approach for learning compact boosted feature descriptors called Low-Dimensional Boosted Gradient Maps (L-BGM). First, we present a modified similarity function well suited for learning low-dimensional, discriminative embeddings with boosting. Next, we show how we can factorize the learned embedding to form a compact feature descriptor. Finally, the gradient-based weak learners utilized by our approach are detailed.

### 3.1 Similarity measure

To mitigate the potentially redundant embeddings found by boosting we propose an alternative similarity function that models the correlation between weak response functions,

$$f_{LBGM}(\mathbf{x}, \mathbf{y}) = \sum_{i,j} \alpha_{i,j} h_i(\mathbf{x}) h_j(\mathbf{y}) = \mathbf{h}(\mathbf{x})^T \mathbf{A} \mathbf{h}(\mathbf{y}), \tag{4}$$

where $\mathbf{h}(\mathbf{x}) = [h_1(\mathbf{x}), \cdots, h_M(\mathbf{x})]$ and $\mathbf{A}$ is an $M \times M$ matrix of coefficients $\alpha_{i,j}$. This similarity measure is a generalization of Equation (2). In particular, $f_{LBGM}$ is equivalent to the Boosted SSC similarity measure in the restricted case of a diagonal $\mathbf{A}$.

Substituting the above expression into Equation (1) gives

$$\mathcal{L}_{LBGM} = \sum_{k=1}^{N} \exp\left( -l_k \sum_{i,j} \alpha_{i,j} h_i(\mathbf{x}_k) h_j(\mathbf{y}_k) \right). \tag{5}$$

Although it can be shown that $\mathcal{L}_{LBGM}$ can be jointly optimized for $\mathbf{A}$ and the $h_i$'s using boosting, this involves a fairly complex procedure. Instead, we propose a two step learning strategy whereby we first apply AdaBoost to find the $h_i$'s as in [3]. As shown by our experiments, this provides an effective way to select relevant $h_i$'s. We then apply stochastic gradient descent to find an optimal weighting over the selected features that minimizes $\mathcal{L}_{LBGM}$.

More formally, let $P$ be the number of relevant response functions found with AdaBoost with $P \ll M$. We define $\mathbf{A}_P \in \mathcal{R}^{P \times P}$ to be the sub-matrix corresponding to the non-zero entries of $\mathbf{A}$, explicitly optimized by our approach. Note that as the loss function is convex in $\mathbf{A}$, $\mathbf{A}_P$ can be found optimally with respect to the selected $h_i$'s. In addition, we constrain $\alpha_{i,j} = \alpha_{j,i}$ during optimization restricting the solution to the set of symmetric $P \times P$ matrices yielding a symmetric similarity measure $f_{LBGM}$. We also experimented with more restrictive forms of regularization, *e.g.*, constraining $\mathbf{A}_P$ to be possitive semi-definite, however, this is more costly and gave similar results.

We use a simple implementation of stochastic gradient descent with a constant valued step size, initialized using the diagonal matrix found by Boosted SSC, and iterate until convergence or a maximum number of iterations is reached. Note that because the weak learners are binary, we can precompute the exponential terms involved in the derivatives for all the data samples, as they are constant with respect to $\mathbf{A}_P$. This significantly speeds up the optimization process.

### 3.2 Embedding factorization

The similarity function of Equation (4) defines an implicit feature mapping over example pairs. We now show how the $\mathbf{A}_P$ matrix in $f_{LBGM}$ can be factorized to result in compact feature descriptors computed independently over each input.

Assuming $\mathbf{A}_P$ to be a symmetric $P \times P$ matrix it can be factorized into the following form,

$$\mathbf{A}_P = \mathbf{B}\mathbf{W}\mathbf{B}^T = \sum_{k=1}^{d} w_k \mathbf{b}_k \mathbf{b}_k^T \tag{6}$$

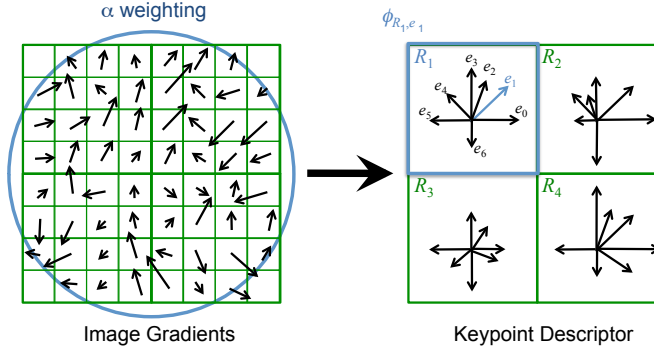

Image Gradients                    Keypoint Descriptor

Figure 1: A specialized configuration of weak response functions $\phi$ corresponding to a regular gridding within the image patch. In addition, assuming a Gaussian weighting of the $\alpha$'s results in a descriptor that closely resembles SIFT [1] and is one of the many solutions afforded by our learning framework.

where $\mathbf{W} = \text{diag}([w_1, \cdots, w_d])$, $w_k \in \{-1, 1\}$, $\mathbf{B} = [\mathbf{b}_1, \cdots, \mathbf{b}_d]$, $\mathbf{b} \in \mathcal{R}^P$, and $d \leq P$.

Equation (4) can then be re-expressed as

$$f_{LBGM}(\mathbf{x}, \mathbf{y}) = \sum_{k=1}^{d} w_k \left( \sum_{i=1}^{P} b_{k,i} h_i(\mathbf{x}) \right) \left( \sum_{j=1}^{P} b_{k,j} h_j(\mathbf{y}) \right) . \tag{7}$$

This factorization defines a signed inner product between the embedded feature vectors and provides increased efficiency with respect to the original similarity measure [1]. For $d < P$ (i.e., the effective rank of $\mathbf{A}_P$ is $d < P$) the factorization represents a smoothed version of $\mathbf{A}_P$ discarding the low-energy dimensions that typically correlate with noise, leading to further performance improvements.

The final embedding found with our approach is therefore

$$H_{LBGM}(\mathbf{x}) = \mathbf{B}^T \mathbf{h}(\mathbf{x}) , \tag{8}$$

and $H_{LBGM}(\mathbf{x}) : \mathcal{R}^D \rightarrow \mathcal{R}^d$.

The projection matrix $\mathbf{B}$ defines a discriminative dimensionality reduction optimized with respect to the exponential loss objective of Equation (5). As seen in our experiments, in the case of redundant $h_i$ this results in a considerable feature compression, also offering a more compact description than the original input patch.

### 3.3 Weak learners

The boosting-trick allows for a variety of non-linear embeddings parameterized by the chosen weak learner family. We employ the gradient-based response functions of [12] to form our feature descriptor. In [12], the usefulness of these features was demonstrated for visual object detection. In what follows, we extend these features to the descriptor matching task illustrating their close connection with the well-known SIFT descriptor.

Following the notation of [12], our weak learners are defined as

$$h(\mathbf{x}; R, e, T) = \begin{cases} 1 & \text{if } \phi_{R,e}(\mathbf{x}) \leq T \\ -1 & \text{otherwise} \end{cases} , \tag{9}$$

where

$$\phi_{R,e}(\mathbf{x}) = \sum_{m \in R} \xi_e(\mathbf{x}, m) \Big/ \sum_{e_k \in \Phi, m \in R} \xi_{e_k}(\mathbf{x}, m) , \tag{10}$$

with region $\xi_e(\mathbf{x}, m)$ being the gradient energy along an orientation $e$ at location $m$ within $\mathbf{x}$, and $R$ defining a rectangular extent within the patch. The gradient energy is computed based on the dot product between $e$ and the gradient orientation at pixel $m$ [12]. The orientation $e$ ranges between $[-\pi, \pi]$ and is quantized to take values $\Phi = \{0, \frac{2\pi}{q}, \frac{4\pi}{q}, \cdots, (q-1) * \frac{2\pi}{q}\}$ with $q$ the number of

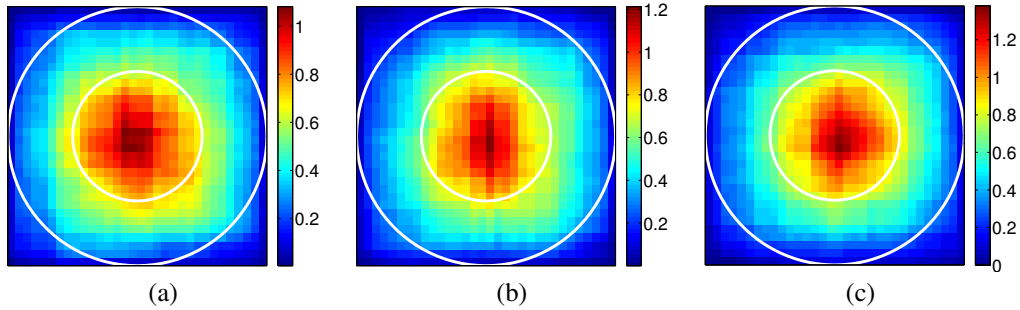

Figure 2: Learned spatial weighting obtained with Boosted Gradient Maps (BGM) trained on (a) Liberty, (b) Notre Dame and (c) Yosemite datasets. The learned weighting closely resembles the Gaussian weighting employed by SIFT (white circles indicate $\sigma/2$ and $\sigma$ used by SIFT).

quantization bins. As noted in [12] this representation can be computed efficiently using integral images.

The non-linear gradient response functions $\phi_{R,e}$ along with their thresholding $T$ define the parameterization of the weak learner family optimized with our approach. Consider the specialized configuration illustrated in Figure 1. This corresponds to a selection of weak learners whose $R$ and $e$ values are parameterized such that they lie along a regular grid, equally sampling each edge orientation within each grid cell. In addition, if we assume a Gaussian weighting centered about the patch, the resulting descriptor closely resembles SIFT[2] [1]. In fact, this configuration and weighting corresponds to one of the many solutions afforded by our approach. In [4], they note the importance of allowing for alternative pooling and feature selection strategies, both of which are effectively optimized within our framework. As seen in our experiments, this results in a significant performance gain over hand-designed SIFT.

## 4    Results

In this section, we first present an overview of our evaluation framework. We then show the results obtained using Boosted SSC combined with gradient-based weak learners described in Sec. 3.3. We continue with the results generated when applying the factorized embedding of the matrix $\mathbf{A}$. Finally, we present a comparison of our final descriptor with the state of the art.

### 4.1    Evaluation framework

We evaluate the performance of our methods using three publicly available datasets: Liberty, Notre Dame and Yosemite [4]. Each of them contain over 400k scale- and rotation-normalized $64 \times 64$ patches. These patches are sampled around interest points detected using Difference of Gaussians and the correspondences between patches are found using a multi-view stereo algorithm. The datasets created this way exhibit substantial perspective distortion and various lighting conditions. The ground truth available for each of these datasets describes 100k, 200k and 500k pairs of patches, where 50% correspond to match pairs, and 50% to non-match pairs. In our evaluation, we separately consider each dataset for training and use the held-out datasets for testing. We report the results of the evaluation in terms of ROC curves and 95% error rate as is done in [4].

### 4.2    Boosted Gradient Maps

To show the performance boost we get by using gradient-based weak learners in our boosting scheme, we plot the results for the original Boosted SSC method [3], which relies on thresholded pixel intensities as weak learners, and for the same method which uses gradient-based weak learners instead (referred to as Boosted Gradient Maps (BGM)) with $q = 24$ quantized orientation bins used throughout our experiments. As we can see in Fig. 3(a), a 128-dimensional Boosted SSC descriptor can be easily outperformed by a 32-dimensional BGM descriptor. When comparing descriptors with the same dimensionality, the improvement measured in terms of 95% error rate reaches over 50%. Furthermore, it is worth noticing, that with 128 dimensions BGM performs similarly to SIFT, and when we increase the dimensionality to 512 - it outperforms SIFT by 14% in terms of 95% error rate. When comparing the 256-dimensional SIFT (obtained by increasing the granularity of the orientation bins) with the 256-dimensional BGM, the extended SIFT descriptor performs much worse

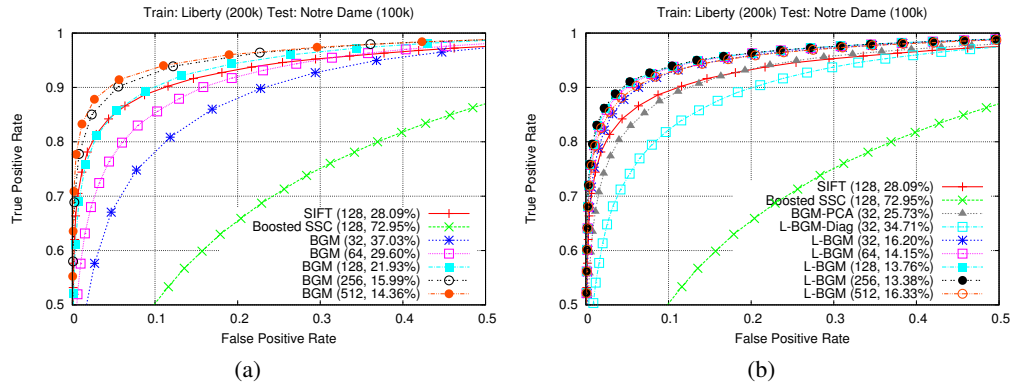

Figure 3: **(a)** Boosted SCC using thresholded pixel intensities in comparison with our Boosted Gradient Maps (BGM) approach. **(b)** Results after optimization of the correlation matrix **A**. Performance is evaluated with respect to factorization dimensionality $d$. In parentheses: the number of dimensions and the 95% error rate.

(34.22% error rate vs 15.99% for BGM-256). This indicates that boosting with a similar number of non-linear classifiers adds to the performance, and proves how well tuned the SIFT descriptor is.

Visualizations of the learned weighting obtained with BGM trained on Liberty, Notre Dame and Yosemite datasets are displayed in Figure 2. To plot the visualizations we sum the $\alpha$'s across orientations within the rectangular regions of the corresponding weak learners. Note that although there are some differences, interestingly this weighting closely resembles the Gaussian weighting employed by SIFT.

### 4.3 Low-Dimensional Boosted Gradient Maps

To further improve performance, we optimize over the correlation matrix of the weak learners' responses, as explained in Sec. 3.1, and apply the embedding from Sec. 3.2. The results of this method are shown in Fig. 3(b). In these experiments, we learn our L-BGM descriptor using the responses of 512 gradient-based weak learners selected with boosting. We first optimize over the weak learners' correlation matrix which is constrained to be diagonal. This corresponds to a global optimization of the weights of the weak learners. The resulting 32-dimensional L-BGM-Diag descriptor performs only slightly better than the corresponding 32-dimensional BGM. Interestingly, the additional degrees of freedom obtained by optimizing over the full correlation matrix boost the results significantly and allow us to outperform SIFT with as few as 32 dimensions. When we compare our 128-dimensional descriptor, *i.e.*, the descriptor of the same length as SIFT, we observe 15% improvement in terms of 95% error rate. However, when we increase the descriptor length from 256 to 512 we can see a slight performance drop since we begin to include the "noisy" dimensions of our embedding which correspond to the eigenvalues of low magnitude, a trend typical to many dimensionality reduction techniques. Hence, as our final descriptor, we select the 64-dimensional L-BGM descriptor, as it provides a decent trade-off between performance and descriptor length.

Figure 3(b) also shows the results obtained by applying PCA on the responses of 512 gradient-based weak learners (BGM-PCA). The descriptor generated this way performs similarly to SIFT, however our method still provides better results even for the same dimensionality, which shows the advantage in optimizing the exponential loss of Eq. 5.

### 4.4 Comparison with the state of the art

Here we compare our approach against the following baselines: sum of squared differences of pixel intensities (SSD), the state-of-the-art SIFT descriptor [1], SURF descriptor [2], binary LDAHash descriptor [5], a real-valued descriptor computed by applying LDE projections on bias-gain normalized patches (LDA-int) [4] and the original Boosted SSC [3]. We have also tested recent binary descriptors such as BRIEF [27], ORB [28] or BRISK [29], however, they performed much worse than the baselines presented in the paper. For SIFT, we use the publicly available implementation of A. Vedaldi [30]. For SURF and LDAHash, we use the implementation available from the websites of the authors. For the other methods, we use our own implementation. For LDA-int we choose the dimensionality which was reported to perform the best on a given dataset according to [4]. For Boosted SSC, we use 128-dimensions as this obtained the best performance.

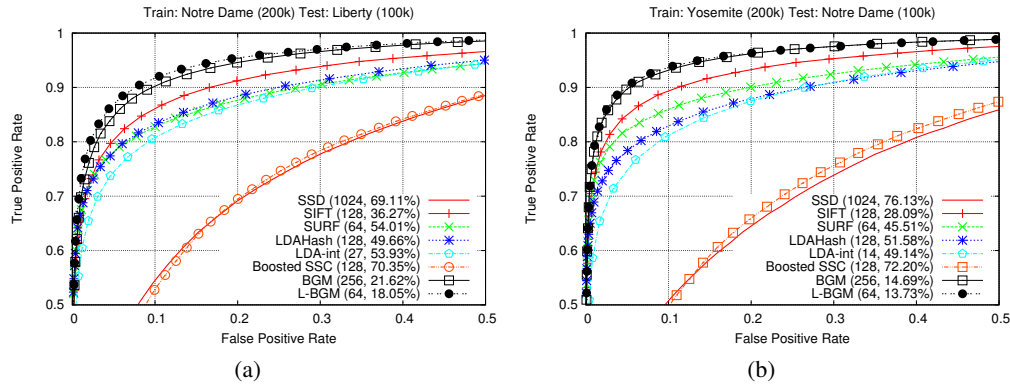

Figure 4: Comparison to state of the art. In parentheses: the number of dimensions, and the 95% error rate. Our L-BGM approach outperforms SIFT by up to 18% in terms of 95% error rate using half fewer dimensions.

In Fig. 4 we plot the recognition curves for all the baselines and our method. BGM and L-BGM outperform the baseline methods across all FP rates. The maximal performance boost is obtained by using our 64-dimensional L-BGM descriptor that results in an up to 18% improvement in terms of 95% error rate with respect to the state-of-the-art SIFT descriptor. Descriptors derived from patch intensities, *i.e.* SSD, Boosted SSC and LDA-int, perform much worse than the gradient-based ones. Finally, our BGM and L-BGM descriptors far outperform SIFT which relies on hand-crafted filters applied to gradient maps. Moreover, with BGM and L-BGM we are able to reduce the 95% error rate by over 3 times with respect to the other state-of-the-art descriptors, namely SURF and LDAHash. We have computed the results for all the configurations of training and testing datasets without observing any significant differences, thus we show here only a representative set of the curves. More results can be found in the supplementary material.

Interestingly, the results we obtain are comparable with "the best of the best" results reported in [4]. However, since the code for their compact descriptors is not publicly available, we can only compare the performance in terms of the 95% error rates. Only the composite descriptors of [4] provide some advantage over our compact L-BGM, as their average 95% error rate is 2% lower than this of L-BGM. Nevertheless, we outperform their non-parametric descriptors by 12% and perform slightly better than the parametric ones, while using descriptors of an order of magnitude shorter. This comparison indicates that even though our approach does not require any complex pipeline optimization and parameter tuning, we perform similarly to the finely optimized descriptors presented in [4].

## 5 Conclusions

In this paper we presented a new method for learning image descriptors by using Low-Dimensional Boosted Gradient Maps (L-BGM). L-BGM offers an attractive alternative to traditional descriptor learning techniques that model non-linearities based on the kernel-trick, relying on a pre-specified kernel function whose selection can be difficult and unintuitive. In contrast, we have shown that for the descriptor matching problem the boosting-trick leads to non-linear feature mappings whose features have an intuitive explanation. We demonstrated the use of gradient-based weak learner functions for learning descriptors within our framework, illustrating their close connection with the well-known SIFT descriptor. A discriminative embedding technique was also presented, yielding fairly compact and discriminative feature descriptions compared to the baseline methods. We evaluated our approach on benchmark datasets where L-BGM was shown to outperform leading contemporary hand-designed and learned feature descriptors. Unlike previous approaches, our L-BGM descriptor can be learned directly from raw intensity patches achieving state-of-the-art performance. Interesting avenues of future work include the exploration of other weak learner families for descriptor learning, e.g., SURF-like Haar features, and extensions to binary feature embeddings.

### Acknowledgments

We would like to thank Karim Ali for sharing his feature code and his insightful feedback and discussions.

## Footnotes

[1]Matching two sets of descriptors each of size $N$ is $\mathcal{O}(N^2 P^2)$ under the original measure and $\mathcal{O}(NPd + N^2 d)$ provided the factorization, resulting in significant savings for reasonably sized $N$ and $P$, and $d \ll P$.

[2]SIFT additionally normalizes each descriptor to be unit norm, however, the underlying representation is otherwise quite similar.

## References

[1] Lowe, D.: Distinctive Image Features from Scale-Invariant Keypoints. IJCV **20**(2) (2004) 91–110

[2] Bay, H., Tuytelaars, T., Van Gool, L.: SURF: Speeded Up Robust Features. In: ECCV'06

[3] Shakhnarovich, G.: Learning Task-Specific Similarity. PhD thesis, MIT (2006)

[4] Brown, M., Hua, G., Winder, S.: Discriminative Learning of Local Image Descriptors. PAMI (2011)

[5] Strecha, C., Bronstein, A., Bronstein, M., Fua, P.: LDAHash: Improved Matching with Smaller Descriptors. PAMI **34**(1) (2012)

[6] Kulis, B., Jain, P., Grauman, K.: Fast Similarity Search for Learned Metrics. PAMI (2009) 2143–2157

[7] Shen, C., Kim, J., Wang, L., van den Hengel, A.: Positive Semidefinite Metric Learning with Boosting. In: NIPS. (2009)

[8] Jain, P., Kulis, B., Davis, J., Dhillon, I.: Metric and Kernel Learning using a Linear Transformation. JMLR (2012)

[9] Bi, J., Wu, D., Lu, L., Liu, M., Tao, Y., Wolf, M.: AdaBoost on Low-Rank PSD Matrices for Metric Learning. In: CVPR. (2011)

[10] Viola, P., Jones, M.: Rapid Object Detection Using a Boosted Cascade of Simple Features. In: CVPR'01

[11] Chapelle, O., Shivaswamy, P., Vadrevu, S., Weinberger, K., Zhang, Y., Tseng, B.: Boosted Multi-Task Learning. Machine Learning (2010)

[12] Ali, K., Fleuret, F., Hasler, D., Fua, P.: A Real-Time Deformable Detector. PAMI **34**(2) (2012) 225–239

[13] Dalal, N., Triggs, B.: Histograms of Oriented Gradients for Human Detection. In: CVPR'05

[14] Weiss, Y., Torralba, A., Fergus, R.: Spectral Hashing. NIPS **21** (2009) 1753–1760

[15] Kulis, B., Darrell, T.: Learning to Hash with Binary Reconstructive Embeddings. In: NIPS'09

[16] Salakhutdinov, R., Hinton, G.: Learning a Nonlinear Embedding by Preserving Class Neighbourhood Structure. In: International Conference on Artificial Intelligence and Statistics. (2007)

[17] Salakhutdinov, R., Hinton, G.: Semantic Hashing. International Journal of Approximate Reasoning (2009)

[18] Grauman, K., Darrell, T.: The Pyramid Match Kernel: Discriminative Classification with Sets of Image Features. In: ICCV'05

[19] Shen, C., Welsh, A., Wang, L.: PSDBoost: Matrix Generation Linear Programming for Positive Semidefinite Matrices Learning. In: NIPS. (2008)

[20] Jia, Y., Huang, C., Darrell, T.: Beyond Spatial Pyramids: Receptive Field Learning for Pooled Image Features. In: CVPR'12

[21] Simonyan, K., Vedaldi, A., Zisserman, A.: Descriptor Learning Using Convex Optimisation. In: ECCV'12

[22] Dollár, P., Tu, Z., Perona, P., Belongie, S.: Integral Channel Features. In: BMVC'09

[23] Torralba, A., Fergus, R., Weiss, Y.: Small Codes and Large Databases for Recognition. In: CVPR'08

[24] Ali, K., Fleuret, F., Hasler, D., Fua, P.: A Real-Time Deformable Detector. PAMI (2011)

[25] Freund, Y., Schapire, R.: A Decision-Theoretic Generalization of On-Line Learning and an Application to Boosting. In: European Conference on Computational Learning Theory. (1995)

[26] Rosset, S., Zhu, J., Hastie, T.: Boosting as a Regularized Path to a Maximum Margin Classifier. JMLR (2004)

[27] Calonder, M., Lepetit, V., Ozuysal, M., Trzcinski, T., Strecha, C., Fua, P.: BRIEF: Computing a Local Binary Descriptor Very Fast. PAMI **34**(7) (2012) 1281–1298

[28] Rublee, E., Rabaud, V., Konolidge, K., Bradski, G.: ORB: An Efficient Alternative to SIFT or SURF. In: ICCV'11

[29] Leutenegger, S., Chli, M., Siegwart, R.: BRISK: Binary Robust Invariant Scalable Keypoints. In: ICCV'11

[30] Vedaldi, A.: `http://www.vlfeat.org/~vedaldi/code/siftpp.html`

